# Homeostatic plasticity in Bayesian spiking networks as Expectation Maximization with posterior constraints

**Stefan Habenschuss**\*, **Johannes Bill**\*, **Bernhard Nessler**
Institute for Theoretical Computer Science, Graz University of Technology
{habenschuss,bill,nessler}@igi.tugraz.at

## Abstract

Recent spiking network models of Bayesian inference and unsupervised learning frequently assume either inputs to arrive in a special format or employ complex computations in neuronal activation functions and synaptic plasticity rules. Here we show in a rigorous mathematical treatment how homeostatic processes, which have previously received little attention in this context, can overcome common theoretical limitations and facilitate the neural implementation and performance of existing models. In particular, we show that homeostatic plasticity can be understood as the enforcement of a 'balancing' posterior constraint during probabilistic inference and learning with Expectation Maximization. We link homeostatic dynamics to the theory of variational inference, and show that nontrivial terms, which typically appear during probabilistic inference in a large class of models, drop out. We demonstrate the feasibility of our approach in a spiking Winner-Take-All architecture of Bayesian inference and learning. Finally, we sketch how the mathematical framework can be extended to richer recurrent network architectures. Altogether, our theory provides a novel perspective on the interplay of homeostatic processes and synaptic plasticity in cortical microcircuits, and points to an essential role of homeostasis during inference and learning in spiking networks.

## 1   Introduction

Experimental findings from neuro- and cognitive sciences have led to the hypothesis that humans create and maintain an internal model of their environment in neuronal circuitry of the brain during learning and development [1, 2, 3, 4], and employ this model for Bayesian inference in everyday cognition [5, 6]. Yet, how these computations are carried out in the brain remains largely unknown. A number of innovative models has been proposed recently which demonstrate that in principle, spiking networks can carry out quite complex probabilistic inference tasks [7, 8, 9, 10], and even learn to adapt to their inputs near optimally through various forms of plasticity [11, 12, 13, 14, 15]. Still, in network models for concurrent online inference and learning, most approaches introduce distinct assumptions: Both [12] in a spiking Winner-take-all (WTA) network, and [15] in a rate based WTA network, identified the limitation that inputs must be normalized before being presented to the network, in order to circumvent an otherwise nontrivial (and arguably non-local) dependency of the intrinsic excitability on all afferent synapses of a neuron. Nessler et al. [12] relied on population coded input spike trains; Keck et al. [15] proposed feed-forward inhibition as a possible neural mechanism to achieve this normalization. A theoretically related issue has been encountered by Deneve [7, 11], in which inference and learning is realized in a two-state Hidden Markov Model by a single spiking neuron. Although synaptic learning rules are found to be locally computable, the learning update for intrinsic excitabilities remains intricate. In a different approach, Brea et al. [13] have recently proposed a promising model for Bayes optimal sequence learning in spiking networks

---

in which a global reward signal, which is computed from the network state and synaptic weights, modulates otherwise purely local learning rules. Also the recent innovative model for variational learning in recurrent spiking networks by Rezende et al. [14] relies on sophisticated updates of variational parameters that complement otherwise local learning rules.

There exists great interest in developing Bayesian spiking models which require minimal non-standard neural mechanisms or additional assumptions on the input distribution: such models are expected to foster the analysis of biological circuits from a Bayesian perspective [16], and to provide a versatile computational framework for novel neuromorphic hardware [17]. With these goals in mind, we introduce here a novel theoretical perspective on homeostatic plasticity in Bayesian spiking networks that complements previous approaches by constraining statistical properties of the *network response* rather than the input distribution. In particular we introduce *'balancing' posterior constraints* which can be implemented in a purely local manner by the spiking network through a simple rule that is strongly reminiscent of homeostatic intrinsic plasticity in cortex [18, 19]. Importantly, it turns out that the emerging network dynamics eliminate a particular class of nontrivial computations that frequently arise in Bayesian spiking networks.

First we develop the mathematical framework for Expectation Maximization (EM) with homeostatic posterior constraints in an instructive Winner-Take-all network model of probabilistic inference and unsupervised learning. Building upon the theoretical results of [20], we establish a rigorous link between homeostatic intrinsic plasticity and variational inference. In a second step, we sketch how the framework can be extended to recurrent spiking networks; by introducing posterior constraints on the correlation structure, we recover local plasticity rules for recurrent synaptic weights.

## 2 Homeostatic plasticity in WTA circuits as EM with posterior constraints

We first introduce, as an illustrative and representative example, a generative mixture model $p(\boldsymbol{z}, \boldsymbol{y}|\boldsymbol{V})$ with hidden causes $\boldsymbol{z}$ and binary observed variables $\boldsymbol{y}$, and a spiking WTA network $\mathcal{N}$ which receives inputs $\boldsymbol{y}(t)$ via synaptic weights $\boldsymbol{V}$. As shown in [12], such a network $\mathcal{N}$ can implement probabilistic inference $p(\boldsymbol{z}|\boldsymbol{y}, \boldsymbol{V})$ through its spiking dynamics, and maximum likelihood learning through local synaptic learning rules (see Figure 1A). The mixture model comprises $K$ binary and mutually exclusive components $z_k \in \{0, 1\}$, $\sum_{k=1}^{K} z_k = 1$, each specialized on a different $N$-dimensional input pattern:

$$p(\boldsymbol{y}, \boldsymbol{z}|\boldsymbol{V}) = \prod_{k=1}^{K} e^{\hat{b}_k z_k} \prod_{i=1}^{N} \left[ (\pi_{ki})^{y_i} \cdot (1 - \pi_{ki})^{1-y_i} \right]^{z_k} \tag{1}$$

$$\Leftrightarrow \log p(\boldsymbol{y}, \boldsymbol{z}|\boldsymbol{V}) = \sum_k z_k \left( \sum_i V_{ki} y_i - A_k + \hat{b}_k \right) \ , \tag{2}$$

$$\text{with } \sum_k e^{\hat{b}_k} = 1 \text{ and } \pi_{ki} = \sigma(V_{ki}) \text{ and } A_k = \sum_i \log(1 + e^{V_{ki}}) \ , \tag{3}$$

where $\sigma(x) = (1 + \exp(-x))^{-1}$ denotes the logistic function, and $\pi_{ki}$ the expected activation of input $i$ under the mixture component $k$. For simplicity and notational convenience, we will treat the prior parameters $\hat{b}_k$ as constants throughout the paper. Probabilistic inference of hidden causes $z_k$ based on an observed input $\boldsymbol{y}$ can be implemented by a spiking WTA network $\mathcal{N}$ of $K$ neurons which fire with the instantaneous spiking probability (for $\delta t \to 0$),

$$p(z_k \text{ spikes in } [t, t+\delta t]) = \delta t \cdot r_{\text{net}} \cdot \frac{e^{u_k(t)}}{\sum_j e^{u_j(t)}} \propto p(z_k = 1|\boldsymbol{y}, \boldsymbol{V}) \ , \tag{4}$$

with the input potential $u_k(t) = \sum_i V_{ki} y_i(t) - A_k + \hat{b}_k$. Each WTA neuron $k$ receives spiking inputs $y_i$ via synaptic weights $V_{ki}$ and responds with an instantaneous spiking probability which depends exponentially on its input potential $u_k$ in accordance with biological findings [21]. Stochastic winner-take-all (soft-max) competition between the neurons is modeled via divisive normalization (4) [22]. The input is defined as $y_i(t) = 1$ if input neuron $i$ emitted a spike within the last $\tau$ milliseconds, and $0$ otherwise, corresponding to a rectangular post-synaptic potential (PSP) of length $\tau$. We define $z_k(t) = 1$ at spike times $t$ of neuron $k$ and $z_k(t) = 0$ otherwise.

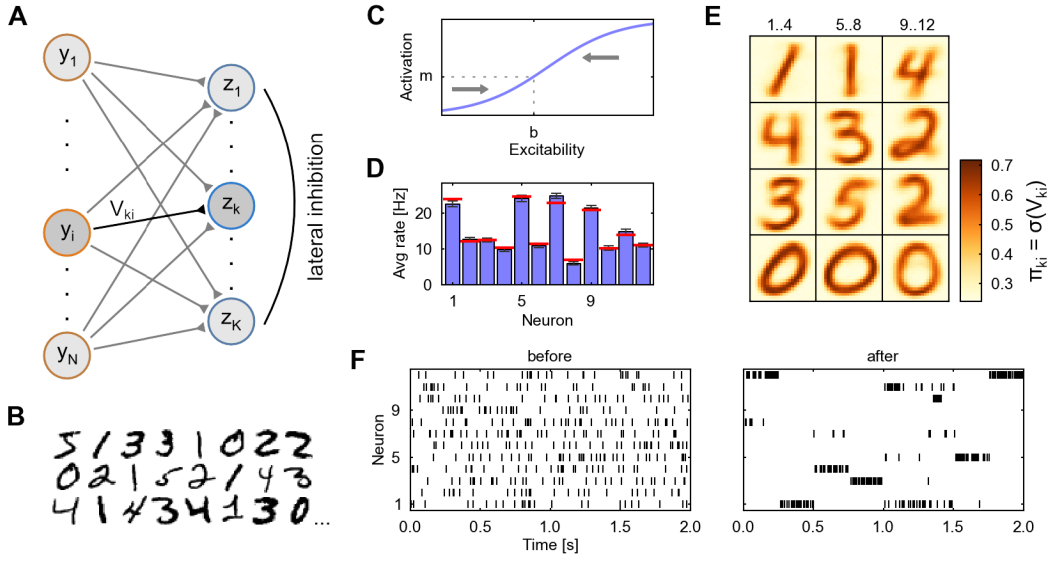

Figure 1: **A**. Spiking WTA network model. **B**. Input templates from MNIST database (digits *0-5*) are presented in random order to the network as spike trains (the input template switches after every 250ms, black/white pixels are translated to high/low firing rates between 20 and 90 Hz). **C**. Sketch of intrinsic homeostatic plasticity maintaining a certain target average activation. **D**. Homeostatic plasticity induces average firing rates (blue) close to target values (red). **E**. After a learning period, each WTA neuron has specialized on a particular input motif. **F**. WTA output spikes during a test phase before and after learning. Learning leads to a sparse output code.

In addition to the spiking input, each neuron's potential $u_k$ features an intrinsic excitability $-A_k + \hat{b}_k$. Note that, besides the prior constant $\hat{b}_k$, this excitability depends on the normalizing term $A_k$, and hence on all afferent synaptic weights through (3): WTA neurons which encode strong patterns with high probabilities $\pi_{ki}$ require lower intrinsic excitabilities, while neurons with weak patterns require larger excitabilities. In the presence of synaptic plasticity, i.e., time-varying $V_{ki}$, it is unclear how biologically realistic neurons could communicate ongoing changes in synaptic weights from distal synaptic sites to the soma. This critical issue was apparently identified in [12] and [15]; both papers circumvent the problem (in similar probabilistic models) by constraining the input $\boldsymbol{y}$ (and also the synaptic weights in [15]) in order to maintain constant and uniform values $A_k$ across all WTA neurons.

Here, we propose a different approach to cope with the nontrivial computations $A_k$ during inference and learning in the network. Instead of assuming that the inputs $\boldsymbol{y}$ meet a normalization constraint, we constrain the *network response* during inference, by applying homeostatic dynamics to the intrinsic excitabilities. This approach turns out to be beneficial in the presence of time-varying synaptic weights, i.e., during ongoing changes of $V_{ki}$ and $A_k$. The resulting interplay of intrinsic and synaptic plasticity can be best understood from the standard EM lower bound [23],

$$F(\boldsymbol{V}, q(\boldsymbol{z}|\boldsymbol{y})) = L(\boldsymbol{V}) - \langle \mathrm{KL}\left(q(\boldsymbol{z}|\boldsymbol{y}) \,||\, p(\boldsymbol{z}|\boldsymbol{y}, \boldsymbol{V})\right\rangle_{p^*(\boldsymbol{y})} \qquad \rightarrow \text{E-step} \;, \qquad (5)$$

$$= \langle \log p(\boldsymbol{y}, \boldsymbol{z}|\boldsymbol{V}) \rangle_{p^*(\boldsymbol{y})q(\boldsymbol{z}|\boldsymbol{y})} + \langle H(q(\boldsymbol{z}|\boldsymbol{y})) \rangle_{p^*(\boldsymbol{y})} \qquad \rightarrow \text{M-step} \;, \qquad (6)$$

where $L(\boldsymbol{V}) = \langle \log p(\boldsymbol{y}|\boldsymbol{V}) \rangle_{p^*(\boldsymbol{y})}$ denotes the log-likelihood of the input under the model, $\mathrm{KL}\left(\cdot \,||\, \cdot\right)$ the Kullback-Leibler divergence, and $H(\cdot)$ the entropy. The decomposition holds for arbitrary distributions $q$. In hitherto proposed neural implementations of EM [11, 12, 15, 24], the network implements the current posterior distribution in the E-step, i.e., $q = p$ and $\mathrm{KL}\left(q \,||\, p\right) = 0$. In contrast, by applying homeostatic plasticity, the network response will be constrained to implement a variational posterior from a class of "homeostatic" distributions $\mathcal{Q}$: the long-term average activation of each WTA neuron $z_k$ is constrained to an a priori defined target value. Notably, we will see that the resulting network response $q^*$ describes an optimal variational E-Step in the sense that $q^*(\boldsymbol{z}|\boldsymbol{y}) = \arg\min_{q \in \mathcal{Q}} \mathrm{KL}\left(q(\boldsymbol{z}|\boldsymbol{y}) \,||\, p(\boldsymbol{z}|\boldsymbol{y}, \boldsymbol{V})\right)$. Importantly, homeostatic plasticity fully regulates the intrinsic excitabilities, and as a side effect eliminates the non-local terms $A_k$ in the E-step,

while synaptic plasticity of the weights $V_{ki}$ optimizes the underlying probabilistic model $p(\boldsymbol{y}, \boldsymbol{z}|\boldsymbol{V})$ in the M-step.

In summary, the network response implements $q^*$ as the variational E-step, the M-Step can be performed via gradient ascent on (6) with respect to $V_{ki}$. As derived in section 2.1, this gives rise to the following temporal dynamics and plasticity rules in the spiking network, which instantiate a stochastic version of the variational EM scheme:

$$u_k(t) = \sum_i V_{ki} y_i(t) + b_k \ , \qquad \dot{b}_k(t) = \eta_b \cdot (r_{\text{net}} \cdot m_k - \delta(z_k(t) - 1)) \ , \tag{7}$$

$$\dot{V}_{ki}(t) = \eta_V \cdot \delta(z_k(t) - 1) \cdot (y_j(t) - \sigma(V_{ki})) \ , \tag{8}$$

where $\delta(\cdot)$ denotes the Dirac delta function, and $\eta_b$, $\eta_V$ are learning rates (which were kept time-invariant in the simulations with $\eta_b = 10 \cdot \eta_V$). Note that (8) is a spike-timing dependent plasticity rule (cf. [12]) and is non-zero only at post-synaptic spike times $t$, for which $z_k(t) = 1$. The effect of the homeostatic intrinsic plasticity rule (7) is illustrated in Figure 1C: it aims to keep the long-term average activation of each WTA neuron $k$ close to a certain target value $m_k$. More precisely, if $r_k$ is a neuron's long-term average firing rate, then homeostatic plasticity will ensure that $r_k / r_{\text{net}} \approx m_k$. The target activations $m_k \in (0, 1)$ can be chosen freely with the obvious constraint that $\sum_k m_k = 1$. Note that (7) is strongly reminiscent of homeostatic intrinsic plasticity in cortex [18, 19].

We have implemented these dynamics in a computer simulation of a WTA spiking network $\mathcal{N}$. Inputs $\boldsymbol{y}(t)$ were defined by translating handwritten digits *0-5* (Figure 1B) from the MNIST dataset [25] into input spike trains. Figure 1D shows that, at the end of a $10^4$s learning period, homeostatic plasticity has indeed achieved that $r_k \approx r_{\text{net}} \cdot m_k$. Figure 1E illustrates the patterns learned by each WTA neuron after this period (shown are the $\pi_{ki}$). Apparently, the WTA neurons have specialized on patterns of different intensity which correspond to different values of $A_k$. Figure 1F shows the output spiking behavior of the circuit before and after learning in response to a set of test patterns. The specialization to different patterns has led to a distinct sparse output code, in which any particular test pattern evokes output spikes from only one or two WTA neurons. Note that homeostasis forces all WTA neurons to participate in the competition, and thus prevents neurons from becoming underactive if their synaptic weights decrease, and from becoming overactive if their synaptic weights increase, much like the original $A_k$ terms (which are nontrivial to compute for the network). Indeed, the learned synaptic parameters and the resulting output behavior corresponds to what would be expected from an optimal learning algorithm for the mixture model (1)-(3).[1]

## 2.1 Theory for the WTA model

In the following, we develop the three theoretical key results for the WTA model (1)-(3):

- Homeostatic intrinsic plasticity finds the network response distribution $q^*(\boldsymbol{z}|\boldsymbol{y}) \in \mathcal{Q}$ closest to the posterior distribution $p(\boldsymbol{z}|\boldsymbol{y}, \boldsymbol{V})$, from a set of "homeostatic" distributions $\mathcal{Q}$.
- The interplay of homeostatic and synaptic plasticity can be understood from the perspective of variational EM.
- The critical non-local terms $A_k$ defined by (3) drop out of the network dynamics.

**E-step: variational inference with homeostasis**

The variational distribution $q(\boldsymbol{z}|\boldsymbol{y})$ we consider for the model (1)-(3) is a $2^N \cdot K$ dimensional object. Since $q$ describes a conditional probability distribution, it is non-negative and normalized for all $\boldsymbol{y}$. In addition, we constrain $q$ to be a "homeostatic" distribution $q \in \mathcal{Q}$ such that the average activation of each hidden variable (neuron) $z_k$ equals an a-priori specified mean activation $m_k$ under the input statistics $p^*(\boldsymbol{y})$. This is sketched in Figure 2. Formally we define the constraint set,

$$\mathcal{Q} = \{q : \langle z_k \rangle_{p^*(\boldsymbol{y}) q(\boldsymbol{z}|\boldsymbol{y})} = m_k, \ \text{for all } k = 1 \ldots K\} \ , \qquad \text{with } \sum_k m_k = 1 \ . \tag{9}$$

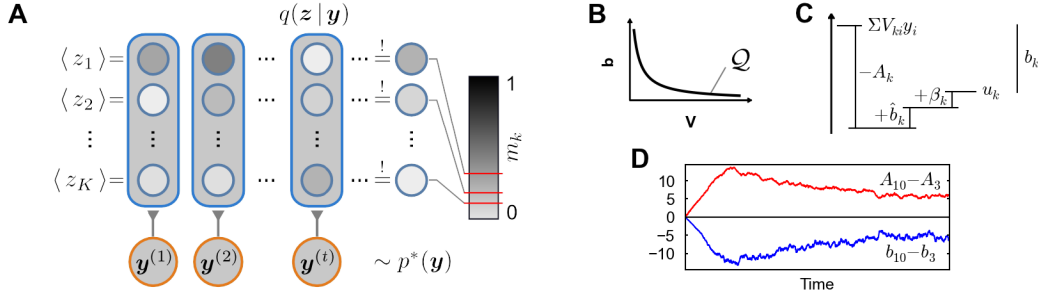

Figure 2: **A**. Homeostatic posterior constraints in the WTA model: Under the variational distribution $q$, the average activation of each variable $z_k$ must equal $m_k$. **B**. For each set of synaptic weights $\boldsymbol{V}$ there exists a unique assignment of intrinsic excitabilities $\boldsymbol{b}$, such that the constraints are fulfilled. **C**. Theoretical decomposition of the intrinsic excitability $b_k$ into $-A_k$, $\hat{b}_k$ and $\beta_k$. **D**. During variational EM the $b_k$ predominantly "track" the dynamically changing non-local terms $-A_k$ (relative comparison between two WTA neurons from Figure 1).

The constrained maximization problem $q^*(\boldsymbol{z}|\boldsymbol{y}) = \arg\max_{q \in \mathcal{Q}} F(\boldsymbol{V}, q(\boldsymbol{z}|\boldsymbol{y}))$ can be solved with the help of Lagrange multipliers (cf. [20]). We find that the $q^*$ which maximizes the objective function $F$ during the E-step (and thus minimizes the KL-divergence to the posterior $p(\boldsymbol{z}|\boldsymbol{y}, \boldsymbol{V})$) has the convenient form $q^*(\boldsymbol{z}|\boldsymbol{y}) \propto p(\boldsymbol{z}|\boldsymbol{y}, \boldsymbol{V}) \cdot \exp(\sum_k \beta_k^* z_k)$ with some $\beta_k^*$. Hence, it suffices to consider distributions of the form,

$$q_{\boldsymbol{\beta}}(\boldsymbol{z}|\boldsymbol{y}) \propto \exp(\sum_k z_k(\sum_i V_{ki} y_i + \underbrace{\hat{b}_k - A_k + \beta_k}_{=:b_k})) \ , \tag{10}$$

for the maximization problem. We identify $\beta_k$ as the variational parameters which remain to be optimized. Note that any distribution of this form can be implemented by the spiking network $\mathcal{N}$ if the intrinsic excitabilities are set to $b_k = -A_k + \hat{b}_k + \beta_k$. The optimal variational distribution $q^*(\boldsymbol{z}|\boldsymbol{y}) = q_{\boldsymbol{\beta}^*}(\boldsymbol{z}|\boldsymbol{y})$ then has $\boldsymbol{\beta}^* = \arg\max_{\boldsymbol{\beta}} \Psi(\boldsymbol{\beta})$, i.e. the variational parameter vector which maximizes the dual [20],

$$\Psi(\boldsymbol{\beta}) = \sum_k \beta_k m_k - \langle \log \sum_{\boldsymbol{z}} p(\boldsymbol{z}|\boldsymbol{y}, \boldsymbol{V}) \exp(\sum_k \beta_k z_k) \rangle_{p^*(\boldsymbol{y})} \ . \tag{11}$$

Due to concavity of the dual, a unique global maximizer $\boldsymbol{\beta}^*$ exists, and thus also the corresponding optimal intrinsic excitabilities $b_k^* = -A_k + \hat{b}_k + \beta_k^*$ are unique. Hence, the posterior constraint $q \in \mathcal{Q}$ can be illustrated as in Figure 2B: For each synaptic weight configuration $\boldsymbol{V}$ there exists, under a particular input distribution $p^*(\boldsymbol{y})$, a unique configuration of intrinsic excitabilities $\boldsymbol{b}$ such that the resulting network output fulfills the homeostatic constraints. The theoretical relation between the intrinsic excitabilities $b_k$, the original nontrivial term $-A_k$ and the variational parameters $\beta_k$ is sketched in Figure 2C. Importantly, while $b_k$ is implemented in the network, $A_k$, $\beta_k$ and $\hat{b}_k$ are not explicitly represented in the implementation anymore. Finding the optimal $\boldsymbol{b}$ in the dual perspective, i.e. those intrinsic excitabilities which fulfill the homeostatic constraints, amounts to gradient ascent $\partial_{\boldsymbol{\beta}} \Psi(\boldsymbol{\beta})$ on the dual, which leads to the following homeostatic learning rule for the intrinsic excitabilities,

$$\Delta b_k \propto \partial_{\beta_k} \Psi(\boldsymbol{\beta}) = m_k - \langle z_k \rangle_{p^*(\boldsymbol{y})q(\boldsymbol{z}|\boldsymbol{y})} \ . \tag{12}$$

Note that the intrinsic homeostatic plasticity rule (7) in the network corresponds to a sample-based stochastic version of this theoretically derived adaptation mechanism (12). Hence, given enough time, homeostatic plasticity will automatically install near-optimal intrinsic excitabilities $\boldsymbol{b} \approx \boldsymbol{b}^*$ and implement the correct variational distribution $q^*$ up to stochastic fluctuations in $\boldsymbol{b}$ due to the non-zero learning rate $\eta_b$. The non-local terms $A_k$ have entirely dropped out of the network dynamics, since the intrinsic excitabilities $b_k$ can be arbitrarily initialized, and are then fully regulated by the local homeostatic rule, which does not require knowledge of $A_k$.

As a side remark, note that although the variational parameters $\beta_k$ are not explicitly present in the implementation, they can be theoretically recovered from the network at any point, via

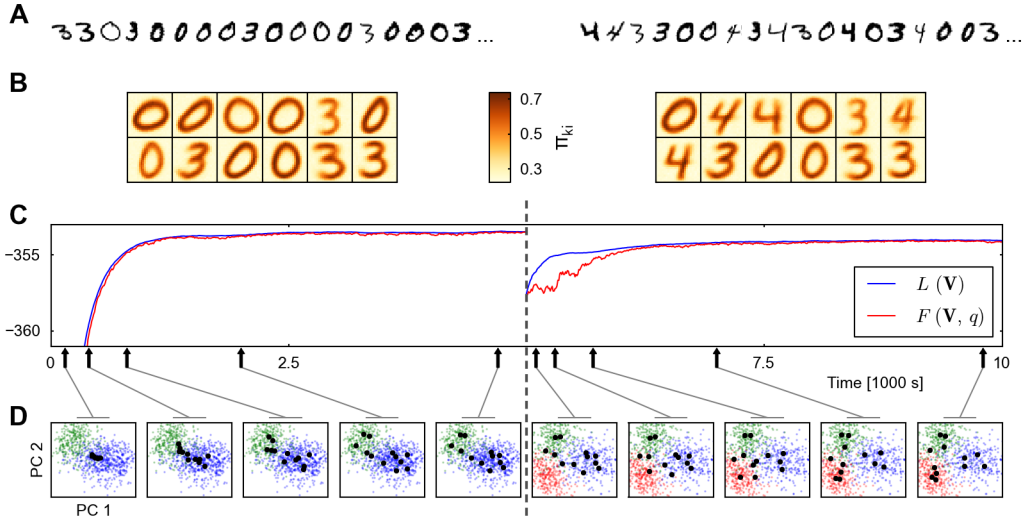

Figure 3: **A**. Input templates from MNIST dataset (digits *0,3* at a ratio 2:1, and digits *0,3,4* at a ratio 1:1:1) used during the first and second learning period, respectively. **B**. Learned patterns at the end of each learning period. **C**. Network performance converges in the course of learning. $F$ is a tight lower bound to $L$. **D**. Illustration of pattern learning and re-learning dynamics in a 2-D projection in the input space. Each black dot corresponds to the pattern $\pi_{ki}$ of one WTA neuron $k$. Colored dots are input samples from the training set (blue/green/red $\leftrightarrow$ digits *0/3/4*).

$\beta_k = b_k + A_k - \hat{b}_k$. Notably, in all our simulations we have consistently found small absolute values of $\beta_k$, corresponding to a small KL-divergence between $q^*$ and $p$.[2] Hence, a major effect of the local homeostatic plasticity rule during learning is to dynamically track and effectively implement the non-local terms $-A_k$. This is shown in Figure 2D, in which the relative excitabilities of two WTA neurons $b_k - b_j$ are plotted against the corresponding non-local $A_k - A_j$ over the course of learning in the first simulation (Figure 1).

**M-step: interplay of synaptic and homeostatic intrinsic plasticity**

During the M-step, we aim to increase the EM lower bound $F$ in (6) w.r.t. the synaptic parameters $\boldsymbol{V}$. Gradient ascent yields,

$$\partial_{V_{ki}} F(\boldsymbol{V}, q(\boldsymbol{z}|\boldsymbol{y})) = \langle \partial_{V_{ki}} \log p(\boldsymbol{y}, \boldsymbol{z}|\boldsymbol{V}) \rangle_{p^*(\boldsymbol{y})q(\boldsymbol{z}|\boldsymbol{y})} \tag{13}$$

$$= \langle z_k \cdot (y_j - \sigma(V_{ki})) \rangle_{p^*(\boldsymbol{y})q(\boldsymbol{z}|\boldsymbol{y})} \ , \tag{14}$$

where $q$ is the variational distribution determined during the E-step, i.e., we can set $q = q^*$. Note the formal correspondence of (14) with the network synaptic learning rule (8). Indeed, if the network activity implements $q^*$, it can be shown easily that the expected update of synaptic weights due to the synaptic plasticity (8) is proportional to (14), and hence implements a stochastic version of the theoretical M-step (cf. [12]).

## 2.2 Dynamical properties of the Bayesian spiking network with homeostasis

To highlight a number of salient dynamical properties emerging from homeostatic plasticity in the considered WTA model, Figure 3 shows a simulation of the same network $\mathcal{N}$ with homeostatic dynamics as in Figure 1, only with different input statistics presented to the network, and uniform $m_k = \frac{1}{K}$. During the first 5000s, different writings of *0*'s and *3*'s from the MNIST dataset were presented, with *0*'s occurring twice as often as *3*'s. Then the input distribution $p^*(\boldsymbol{y})$ abruptly switched to include also *4*'s, with each digit occurring equally often. The following observations can be made: Due to the homeostatic constraint, each neuron responds on average to $m_k \cdot T$ out of $T$ presented inputs. As a consequence, the number of neurons which specialize on a particular digit is

directly proportional to the frequency of occurrence of that digit, i.e. 8:4 and 4:4:4 after the first and second learning period, respectively (Figure 3B). In general, if uniform target activations $m_k$ are chosen, output resources are allocated precisely in proportion to input frequency. Figure 3C depicts the time course of the EM lower bound $F$ as well as the average likelihood $L$ (assuming uniform $\hat{b}_k$) under the model during a single simulation run, demonstrating both convergence and tightness of the lower bound. As expected due to the stabilizing dynamics of homeostasis, we found variability in performance among different trials to be small (not shown). Figure 3D illustrates the dynamics of learning and re-learning of patterns $\pi_{ki}$ in a 2D projection of input patterns onto the first two principal components.

## 3   Homeostatic plasticity in recurrent spiking networks

The neural model so far was essentially a feed-forward network, in which every postsynaptic spike can directly be interpreted as one sample of the instantaneous posterior distribution [12]. The lateral inhibition served only to ensure the normalization of the posterior. We will now extend the concept of homeostatic processes as posterior constraints to the broader class of recurrent networks and sketch the utility of the developed framework beyond the regulation of intrinsic excitabilities.

Recently it was shown in [9, 10] that recurrent networks of stochastically spiking neurons can in principle carry out probabilistic inference through a sampling process. At every point in time, the joint network state $z(t)$ represents one sample of a posterior. However, [9] and [10] did not consider unsupervised learning on spiking input streams.

For the following considerations, we divide the definition of the probabilistic model in two parts. First, we define a Boltzmann distribution,

$$p(\boldsymbol{z}) = \exp(\sum_k \hat{b}_k z_k + \frac{1}{2}\sum_{j \neq k} \hat{W}_{kj} z_k z_j)/\text{norm. } , \qquad (15)$$

with $\hat{W}_{kj} = \hat{W}_{jk}$ as "prior" for the hidden variables $z$ which will be represented by a recurrently connected network of $K$ spiking neurons. For the purpose of this section, we treat $\hat{b}_k$ and $\hat{W}_{kj}$ as constants. Secondly, we define a conditional distribution in the exponential-family form [23],

$$p(\boldsymbol{y}|\boldsymbol{z}, \boldsymbol{V}) = \exp(f_0(\boldsymbol{y}) + \sum_{k,i} V_{ki} z_k f_i(\boldsymbol{y}) - A(\boldsymbol{z}, \boldsymbol{V})) , \qquad (16)$$

that specifies the likelihood of observable inputs $y$, given a certain network state $z$. This defines the generative model $p(\boldsymbol{y}, \boldsymbol{z}|\boldsymbol{V}) = p(\boldsymbol{z})\, p(\boldsymbol{y}|\boldsymbol{z}, \boldsymbol{V})$.

We map this probabilistic model to the spiking network and define that for every $k$ and every point in time $t$ the variable $z_k(t)$ has the value 1, if the corresponding neuron has fired within the time window $(t - \tau, t]$. In accordance with the neural sampling theory, in order for a spiking network to sample from the correct posterior $p(\boldsymbol{z}|\boldsymbol{y}, \boldsymbol{V}) \propto p(\boldsymbol{z})\, p(\boldsymbol{y}|\boldsymbol{z}, \boldsymbol{V})$ given the input $y$, each neuron must compute in its membrane potential the log-odd [9],

$$u_k = \log\frac{p(z_k = 1|\boldsymbol{z}_{\backslash k}, \boldsymbol{V})}{p(z_k = 0|\boldsymbol{z}_{\backslash k}, \boldsymbol{V})} = \underbrace{\sum_i V_{ki} f_i(\boldsymbol{y})}_{\text{feedforward drive}} \underbrace{-A_k(\boldsymbol{V}) + \hat{b}_k}_{\text{intr. excitability}} + \sum_{j \neq k} \underbrace{(-A_{kj}(\boldsymbol{V}) + \hat{W}_{kj})}_{\text{recurrent weight}} z_j - \dots$$

$$(17)$$

where $\boldsymbol{z}_{\backslash k} = (z_1, \dots, z_{k-1}, z_{k+1}, \dots z_K)^{\mathsf{T}}$. The $A_k, A_{kj}, \dots$ are given by the decomposition of $A(\boldsymbol{z}, \boldsymbol{V})$ along the binary combinations of $z$ as,

$$A(\boldsymbol{z}, \boldsymbol{V}) = A_0(\boldsymbol{V}) + \sum_k z_k A_k(\boldsymbol{V}) + \frac{1}{2}\sum_{j \neq k} z_k z_j A_{kj}(\boldsymbol{V}) + \dots \qquad (18)$$

Note, that we do not aim at this point to give learning rules for the prior parameters $\hat{b}_k$ and $\hat{W}_{kj}$. Instead we proceed as in the last section and specify a-priori desired properties of the average network response under the input distribution $p^*(\boldsymbol{y})$,

$$c_{kj} = \langle z_k z_j \rangle_{p^*(\boldsymbol{y})q(\boldsymbol{z}|\boldsymbol{y})} \qquad \text{and} \qquad m_k = \langle z_k \rangle_{p^*(\boldsymbol{y})q(\boldsymbol{z}|\boldsymbol{y})} . \qquad (19)$$

Let us explore some illustrative configurations for $m_k$ and $c_{kj}$. One obvious choice is closely related to the goal of maximizing the entropy of the output code by fixing $\langle z_k \rangle$ to $\frac{1}{K}$ and $\langle z_k z_j \rangle$ to $\langle z_k \rangle \langle z_j \rangle = \frac{1}{K^2}$, thus enforcing second order correlations to be zero. Another intuitive choice would be to set all $\langle z_k z_j \rangle$ very close to zero, which excludes that two neurons can be active simultaneously and thus recovers the function of a WTA. It is further conceivable to assign positive correlation targets to groups of neurons, thereby creating populations with redundant codes. Finally, with a topographical organization of neurons in mind, all three basic ideas sketched above might be combined: one could assign positive correlations to neighboring neurons in order to create local cooperative populations, mutual exclusion at intermediate distance, and zero correlation targets between distant neurons.

With this in mind, we can formulate the goal of learning for the network in the context of EM with posterior constraints: we constrain the E-step such that the average posterior fulfills the chosen targets, and adapt the forward weights $V$ in the M-step according to (6). Analogous to the first-order case, the variational solution of the E-step under these constraints takes the form,

$$ q_{\boldsymbol{\beta},\boldsymbol{\omega}}(\boldsymbol{z}|\boldsymbol{y}) \propto p(\boldsymbol{z}|\boldsymbol{y},\boldsymbol{V}) \cdot \exp\left(\sum_k \beta_k z_k + \frac{1}{2}\sum_{j \neq k} \omega_{kj} z_k z_j\right) \quad, \tag{20} $$

with symmetric $\omega_{kl} = \omega_{lk}$ as variational parameters. A neural sampling network $\mathcal{N}$ with input weights $V_{ki}$ will sample from $q_{\boldsymbol{\beta},\boldsymbol{\omega}}$ if the intrinsic excitabilities are set to $b_k = -A_k + \hat{b}_k + \beta_k$, and the symmetric recurrent synaptic weights to $W_{kj} = -A_{kj} + \hat{W}_{kj} + \omega_{kj}$. The variational parameters $\boldsymbol{\beta}, \boldsymbol{\omega}$ (and hence also $\boldsymbol{b}, \boldsymbol{W}$) which optimize the dual problem $\Psi(\boldsymbol{b},\boldsymbol{\omega})$ are uniquely defined and can be found iteratively via gradient ascent. Analogous to the last section, this yields the intrinsic plasticity rule (12) for $b_k$. In addition, we obtain for the recurrent synapses $W_{kj}$,

$$ \Delta W_{kj} \propto c_{kj} - \langle z_k z_j \rangle_{p^*(\boldsymbol{y})q(\boldsymbol{z}|\boldsymbol{y})} \quad, \tag{21} $$

which translates to an anti-Hebbian spike-timing dependent plasticity rule in the network implementation.

For any concrete instantiation of $f_0(\boldsymbol{y})$, $f_i(\boldsymbol{y})$ and $A(\boldsymbol{z},\boldsymbol{V})$ in (16) it is possible to derive learning rules for $V_{ki}$ for the M-step via $\partial_{V_{ki}} F(\boldsymbol{V},q)$. Of course not all models entail local synaptic learning rules. In particular it might be necessary to assume conditional independence of the inputs $\boldsymbol{y}$ given the network state $\boldsymbol{z}$, i.e., $p(\boldsymbol{y}|\boldsymbol{z},\boldsymbol{V}) = \prod_i p(y_i|\boldsymbol{z},\boldsymbol{V})$. Furthermore, in order to fulfill the neural computability condition (17) for neural sampling [9] with a recurrent network of point neurons, it might be necessary to choose $A(\boldsymbol{z},\boldsymbol{V})$ such that terms of order higher than 2 vanish in the decomposition. This can be shown to hold, for example, in a model with conditionally independent Gaussian distributed inputs $y_i$. It is ongoing work to find further biologically realistic network models in the sense of this theory and to assess their computational capabilities through computer experiments.

## 4 Discussion

Complex and non-local computations, which appear during probabilistic inference and learning, arguably constitute one of the cardinal challenges in the development of biologically realistic Bayesian spiking network models. In this paper we have introduced homeostatic plasticity, which to the best of our knowledge had not been considered before in the context of EM in spiking networks, as a theoretically grounded approach to stabilize and facilitate learning in a large class of network models. Our theory complements previously proposed neural mechanisms and provides, in particular, a simple and biologically realistic alternative to the assumptions on the input distribution made in [12] and [15]. Indeed, our results challenge the hypothesis of [15] that feedforward inhibition is critical for correctly learning the structure of the data with biologically plausible plasticity rules. More generally, it turns out that the enforcement of a balancing posterior constraint often simplifies inference in recurrent spiking networks by eliminating nontrivial computations. Our results suggest a crucial role of homeostatic plasticity in the Bayesian brain: to constrain activity patterns in cortex to assist the autonomous optimization of an internal model of the environment.

**Acknowledgments.** Written under partial support by the European Union - projects #FP7-269921 (BrainScaleS), #FP7-216593 (SECO), #FP7-237955 (FACETS-ITN), #FP7-248311 (AMARSi), #FP7-216886 (PASCAL2) - and the Austrian Science Fund FWF #I753-N23 (PNEUMA).

## Footnotes

[1]Without adaptation of intrinsic excitabilities, the network would start performing erroneous inference, learning would reinforce this erroneous behavior, and performance would quickly break down. We have verified this in simulations for the present WTA model: Consistently across trials, a small subset of WTA neurons became dominantly active while most neurons remained silent.

[2]This is assuming for simplicity uniform prior parameters $\hat{b}_k$. Note that a small KL-divergence is in fact often observed during variational EM since $F$, which contains the negative KL-divergence, is being maximized.

## References

[1] K. P. Körding and D. M. Wolpert. Bayesian integration in sensorimotor learning. *Nature*, 427(6971):244–247, 2004.

[2] G. Orban, J. Fiser, R.N. Aslin, and M. Lengyel. Bayesian learning of visual chunks by human observers. *Proceedings of the National Academy of Sciences*, 105(7):2745–2750, 2008.

[3] J. Fiser, P. Berkes, G. Orban, and M. Lengyel. Statistically optimal perception and learning: from behavior to neural representation. *Trends in Cogn. Sciences*, 14(3):119–130, 2010.

[4] P. Berkes, G. Orban, M. Lengyel, and J. Fiser. Spontaneous cortical activity reveals hallmarks of an optimal internal model of the environment. *Science*, 331:83–87, 2011.

[5] T. L. Griffiths and J. B. Tenenbaum. Optimal predictions in everyday cognition. *Psychological Science*, 17(9):767–773, 2006.

[6] D. E. Angelaki, Y. Gu, and G. C. DeAngelis. Multisensory integration: psychophysics, neurophysiology and computation. *Current opinion in neurobiology*, 19(4):452–458, 2009.

[7] S. Deneve. Bayesian spiking neurons I: Inference. *Neural Computation*, 20(1):91–117, 2008.

[8] A. Steimer, W. Maass, and R.J. Douglas. Belief propagation in networks of spiking neurons. *Neural Computation*, 21:2502–2523, 2009.

[9] L. Buesing, J. Bill, B. Nessler, and W. Maass. Neural dynamics as sampling: A model for stochastic computation in recurrent networks of spiking neurons. *PLoS Comput Biol*, 7(11):e1002211, 11 2011.

[10] D. Pecevski, L. Buesing, and W. Maass. Probabilistic inference in general graphical models through sampling in stochastic networks of spiking neurons. *PLoS Comput Biol*, 7(12), 12 2011.

[11] S. Deneve. Bayesian spiking neurons II: Learning. *Neural Computation*, 20(1):118–145, 2008.

[12] B. Nessler, M. Pfeiffer, and W. Maass. STDP enables spiking neurons to detect hidden causes of their inputs. In *Proc. of NIPS 2009*, volume 22, pages 1357–1365. MIT Press, 2010.

[13] J. Brea, W. Senn, and J.-P. Pfister. Sequence learning with hidden units in spiking neural networks. In *Proc. of NIPS 2011*, volume 24, pages 1422–1430. MIT Press, 2012.

[14] D. J. Rezende, D. Wierstra, and W. Gerstner. Variational learning for recurrent spiking networks. In *Proc. of NIPS 2011*, volume 24, pages 136–144. MIT Press, 2012.

[15] C. Keck, C. Savin, and J. Lücke. Feedforward inhibition and synaptic scaling–two sides of the same coin? *PLoS Computational Biology*, 8(3):e1002432, 2012.

[16] Joshua B. Tenenbaum, Charles Kemp, Thomas L. Griffiths, and Noah D. Goodman. How to grow a mind: Statistics, structure, and abstraction. *Science*, 331(6022):1279–1285, 2011.

[17] J. Schemmel, D. Brüderle, A. Grübl, M. Hock, K. Meier, and S. Millner. A wafer-scale neuromorphic hardware system for large-scale neural modeling. *Proc. of ISCAS'10*, pages 1947–1950, 2010.

[18] N.S. Desai, L.C. Rutherford, and G.G. Turrigiano. Plasticity in the intrinsic excitability of cortical pyramidal neurons. *Nature Neuroscience*, 2(6):515, 1999.

[19] A. Watt and N. Desai. Homeostatic plasticity and STDP: keeping a neurons cool in a fluctuating world. *Frontiers in Synaptic Neuroscience*, 2, 2010.

[20] J. Graca, K. Ganchev, and B. Taskar. Expectation maximization and posterior constraints. In *Proc. of NIPS 2007*, volume 20. MIT Press, 2008.

[21] R. Jolivet, A. Rauch, HR Lüscher, and W. Gerstner. Predicting spike timing of neocortical pyramidal neurons by simple threshold models. *Journal of Computational Neuroscience*, 21:35–49, 2006.

[22] E.P. Simoncelli and D.J. Heeger. A model of neuronal responses in visual area MT. *Vision Research*, 38(5):743–761, 1998.

[23] C. M. Bishop. *Pattern Recognition and Machine Learning*. Springer, New York, 2006.

[24] M. Sato. Fast learning of on-line EM algorithm. *Rapport Technique, ATR Human Information Processing Research Laboratories*, 1999.

[25] Y. LeCun, L. Bottou, Y. Bengio, and P. Haffner. Gradient-based learning applied to document recognition. In *Proceedings of the IEEE*, volume 86, pages 2278–2324, 11 1998.

